# Complexity of Decentralized Control: Special Cases

**Martin Allen**
Department of Computer Science
Connecticut College
New London, CT 06320
martin.allen@conncoll.edu

**Shlomo Zilberstein**
Department of Computer Science
University of Massachusetts
Amherst, MA 01003
shlomo@cs.umass.edu

## Abstract

The worst-case complexity of general decentralized POMDPs, which are equivalent to partially observable stochastic games (POSGs) is very high, both for the cooperative and competitive cases. Some reductions in complexity have been achieved by exploiting independence relations in some models. We show that these results are somewhat limited: when these independence assumptions are relaxed in very small ways, complexity returns to that of the general case.

## 1 Introduction

Decentralized and partially observable stochastic decision and planning problems are very common, comprising anything from strategic games of chance to robotic space exploration. In such domains, multiple agents act under uncertainty about both their environment and the plans and actions of others. These problems can be represented as *decentralized partially observable Markov decision processes* (Dec-POMDPs), or the equivalent, *partially observable stochastic games* (POSGs), allowing for precise formulation of solution concepts and success criteria.

Alas, such problems are highly complex. As shown by Bernstein et al. [1, 2], the full, cooperative problem—where all players share the same payoff, and strategies can depend upon entire observed histories—is NEXP-complete. More recently, Goldsmith and Mundhenk [3] showed that the competitive case can be worse: when teamwork is allowed among agents, complexity rises to $\text{NEXP}^{\text{NP}}$ (problems solvable by a NEXP machine employing an NP set as an oracle). Much attention has thus been paid to restricted cases, particularly those where some parts of the system dynamics behave independently. The complexity of finite-horizon Dec-POMDPs goes down—from NEXP to NP—when agents interact only via a joint reward structure, and are otherwise independent. Unfortunately, our new results show that further reduction, based on other combinations of fully or partially independent system dynamics are unlikely, if not impossible.

We show that if the situation were reversed, so that rewards alone are independent, the problem remains NEXP-complete. Further, we consider two other Dec-POMDP sub-classes from the literature: (a) domains where local agent sub-problems are independent except for a (relatively small) number of *event-based interactions*, and (b) those where agents only interact influencing the set of *currently available actions*. As it turns out, both types of problem are NEXP-complete as well—facts previously unknown. (In the latter case, this is a substantial increase in the known upper bound.) These results provide further impetus to devise new tools for the analysis and classification of problem difficulty in decentralized problem solving.

## 2 Basic definitions

The *cooperative, decentralized partially observable Markov decision process* (Dec-POMDP) is a highly general and powerful framework, capable of representing a wide range of real-world problem

domains. It extends the basic POMDP to multiple agents, operating in conjunction based on locally observed information about the world, and collecting a single source of reward.

**Definition 1** (Dec-POMDP). A (Dec-POMDP), $\mathcal{D}$, is specified by a tuple:

$$M = \langle \{\alpha_i\}, S, \{A_i\}, P, \{\Omega_i\}, O, R, T \rangle \tag{1}$$

with individual components as follows:

- Each $\alpha_i$ is an *agent*; $S$ is a finite set of *world states* with a distinguished *initial state* $s^0$; $A_i$ is a finite set of *actions*, $a_i$, available to $\alpha_i$; $\Omega_i$ is a finite set of *observations*, $o_i$, for $\alpha_i$; and $T$ is the (finite or infinite) *time-horizon* of the problem.

- $P$ is the *Markovian state-action transition function*. $P(s, a_1, \ldots, a_n, s')$ is the probability of going from state $s$ to state $s'$, given joint action $\langle a_1, \ldots, a_n \rangle$.

- $O$ is the *joint observation function* for the set of agents, given each state-action transition. $O(a_1, \ldots, a_n, s', o_1, \ldots, o_n)$ is the probability of observing $\langle o_1, \ldots, o_n \rangle$, if joint action $\langle a_1, \ldots, a_n \rangle$ causes a transition to global state $s'$.

- $R$ is the *global reward function*. $R(s, a_1, \ldots, a_n)$ is the reward obtained for performing joint action $\langle a_1, \ldots, a_n \rangle$ when in global state $s$.

The most important sub-instance of the Dec-POMDP model is the *decentralized MDP* (Dec-MDP), where the joint observation tells us everything we need to know about the system state.

**Definition 2** (Dec-MDP). A *decentralized Markov decision process* (Dec-MDP) is a Dec-POMDP that is *jointly fully observable*. That is, there exists a functional mapping, $J : \Omega_1 \times \cdots \times \Omega_n \to S$, such that $O(a_1, \ldots, a_n, s', o_1, \ldots, o_n) \neq 0$ if and only if $J(o_1, \ldots, o_n) = s'$.

In a Dec-MDP, then, the sum total of the individual agent observations provides a complete picture of the state of the environment. It is important to note, however, that this *does not mean* that any individual agent actually possesses this information. Dec-MDPs are still fully decentralized in general, and individual agents cannot count on access to the global state when choosing actions.

**Definition 3** (Policies). A *local policy* for an agent $\alpha_i$ is a mapping from sequences of that agent's observations, $\overline{o_i} = \langle o_i^1, \ldots, o_i^k \rangle$, to its actions, $\pi_i : \Omega_i^\star \to A_i$. A *joint policy for $n$ agents* is a collection of local policies, one per agent, $\pi = \langle \pi_1, \ldots, \pi_n \rangle$.

A solution method for a decentralized problem seeks to find some joint policy that maximizes expected value given the starting state (or distribution over states) of the problem. For complexity purposes, the decision version of the Dec-(PO)MDP problem is to determine whether there exists some joint policy with value greater at least $k$.

## 3 Bernstein's proof of NEXP-completeness

Before establishing our new claims, we briefly review the NEXP-completeness result for finite-horizon Dec-MDPs, as given by Bernstein et al. [1, 2]. First, we note that the upper bound, namely that finite-horizon Dec-POMDPs are in NEXP, will immediately establish the same upper bound for all the problems that we will consider. (While we do not discuss the proof here, full details can be found in the original, or the supplemental materials to this paper, §1.)

**Theorem 1** (Upper Bound). *The finite-horizon, $n$-agent decision problem Dec-POMDP $\in$ NEXP.*

More challenging (and interesting) is establishing lower bounds on these problems, which is performed via our reduction from the known NEXP-complete TILING problem [4, 5]. A TILING problem instance consists of a board size $n$, given concisely in $\log n$ binary bits, a set of tile-types $L = \{t_0, \ldots, t_k\}$, and a collection of binary and vertical compatibility relations between tiles $H, V \subseteq L \times L$. A *tiling* is a mapping of board locations to tile-types, $t : \{0, \ldots, n-1\} \times \{0, \ldots, n-1\} \to L$; such a tiling is *consistent* just in case (i) the origin location of the board receives tile-type 0 ($t(0,0) = tile_0$); and (ii) all adjoint tile assignments are compatible:

$$(\forall x, y) \ \langle t(x, y), t(x+1, y) \rangle \in H \ \& \ \langle t(x, y), t(x, y+1) \rangle \in V.$$

The TILING problem is thus to decide, for a given instance, whether such a consistent tiling exists. Figure 1 shows an example instance and consistent solution.

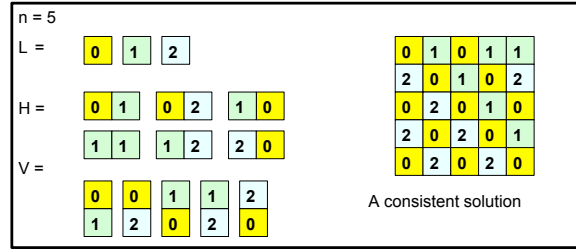

Figure 1: An example of the TILING problem, and a consistent solution.

The reduction transforms a given instance of TILING into a 2-agent Dec-MDP, where each agent is queried about some location in the grid, and must answer with a tile to be placed there. By careful design of the query and response mechanism, it is ensured that a policy with non-negative value exists only if the agents already have a consistent tiling, thus showing the Dec-MDP to be as hard as TILING. Together with Theorem 1, and the fact that the finite-horizon, 2-agent Dec-MDP is a special case of the general finite-horizon Dec-POMDP, the reduction establishes Bernstein's main complexity result (*again, details are in the supplemental materials, §1*):

**Theorem 2** (NEXP-Completeness)**.** The finite-horizon Dec-POMDP problem is NEXP-complete.

## 4    Factored Dec-POMDPs and independence

In general, the state transitions, observations, and rewards in a Dec-POMDP can involve probabilistic dependencies between agents. An obvious restricted subcase is thus one in which these factors are somehow independent. Becker et al. [6, 7] have thus studied problems in which the global state-space consists of the product of local states, so that each agent has its own individual state-space. A Dec-POMDP can then be *transition independent*, *observation independent*, or *reward independent*, as each the local effects given by each corresponding function are independent of one another.

**Definition 4** (Factored Dec-POMDP)**.** A factored, $n$-agent Dec-POMDP is a Dec-POMDP such that the system state can be factored into $n+1$ distinct components, so that $S = S_0 \times S_1 \times \cdots \times S_n$, and no state-variable appears in any $S_i$, $S_j$, $i \neq j$.

As with the *local* (agent-specific) actions, $a_i$, and observations, $o_i$, in the general Dec-POMDP definition, we now refer to the *local state*, $\hat{s} \in S_i \times S_0$, namely that portion of the overall state-space that is either specific to agent $\alpha_i$ ($s_i \in S_i$), or shared among all agents ($s_o \in S_0$). We use the notation $\overline{s}_{-i}$ for the sequence of all state-components *except that* for agent $\alpha_i$:

$$\overline{s}_{-i} = (s_0, \, s_1, \, \ldots, \, s_{i-1}, \, s_{i+1}, \, \ldots, \, s_n)$$

(and similarly for action- or observation-sequences, $\overline{a}_{-i}$ and $\overline{o}_{-i}$).

**Definition 5** (Transition Independence)**.** A factored, $n$-agent DEC-POMDP is *transition independent* iff the state-transition function can be separated into $n+1$ distinct transition functions $P_0, \ldots, P_n$, where, for any next state $s'_i \in S_i$,

$$P(s'_i \,|\, (s_0, \ldots, s_n), (a_1, \ldots, a_n), \overline{s}_{-i}) = \begin{cases} P_0(s'_0 \,|\, s_0) & \text{if } i = 0; \\ P_i(s'_i \,|\, \hat{s}_i, a_i, s'_0) & \text{else.} \end{cases}$$

In other words, the next local state of each agent is independent of the local states of all others, given its previous local state and local action, and the external system features ($S_0$).

**Definition 6** (Observation Independence)**.** A factored, $n$-agent Dec-POMDP is *observation independent* iff the joint observation function can be separated into $n$ separate probability functions $O_1, \ldots, O_n$, where, for any local observation $o_i \in \Omega_i$,

$$O(o_i \,|\, (a_1, \ldots, a_n), (s'_0, \ldots, s'_n), \overline{o}_{-i}) = O_i(o_i \,|\, a_i, \hat{s}'_i)$$

In such cases, the probability of an agent's individual observations is a function of their own local states and actions alone, independent of the states of others, and of what those others do or observe.

**Definition 7** (Reward Independence). A factored, $n$-agent Dec-POMDP is *reward independent* iff the joint reward function can be represented by local reward functions $R_1, \ldots, R_n$, such that:

$$R((s_0, \ldots s_n), (a_0, \ldots, a_n)) = f(R_1(\hat{s}_1, a_1), \ldots, R_n(\hat{s}_n, a_n))$$

and

$$R_i(\hat{s}_i, a_i) \geq R_i(\hat{s}_i, a_i') \ \Leftrightarrow \ f(R_1, \ldots, R_i(\hat{s}_i, a_i), \ldots, R_n) \geq f(R_1, \ldots, R_i(\hat{s}_i, a_i'), \ldots, R_n)$$

That is, joint reward is a function of local reward, constrained so that we maximize global reward if and only if we maximize local rewards. A typical example is the additive sum:

$$R((s_0, \ldots s_n), (a_0, \ldots, a_n)) = R_1(\hat{s}_1, a_1) + \cdots + R_n(\hat{s}_n, a_n).$$

It is important to note that each definition applies equally to Dec-MDPs; in such cases, joint full observability of the *overall state* is often accompanied by full observability at the *local level*.

**Definition 8** (Local Full Observability). A factored, $n$-agent Dec-MDP is *locally fully observable* iff an agent's local observation uniquely determines its local state: $\forall o_i \in \Omega_i, \exists \hat{s}_i : \ P(\hat{s}_i \mid o_i) = 1$.

Local full observability is not equivalent to independence of observations. In particular, a problem may be locally fully observable without being observation independent (since agents may simply observe outcomes of non-independent joint actions). On the other hand, it is easy to show that an observation-independent Dec-MDP must be locally fully observable (*supplementary, §2*).

### 4.1 Shared rewards alone lead to reduced complexity

It is easy to see that if a Dec-MDP (or Dec-POMDP) has all three forms of independence given by Definitions 5–7, it can be decomposed into $n$ separate problems, where each agent $\alpha_i$ works solely within the local sub-environment $S_i \times S_0$. Such single-agent problems are known to be P-complete, and can generally be solved efficiently to high degrees of optimality. More interesting results follow when *only some* forms of independence hold. In particular, it has been shown that Dec-MDPs with both transition- and observation-independence, *but not* reward-independence, are NP-complete [8, 7]. (*This result is discussed in detail in our supplementary material, §3.*)

**Theorem 3.** A transition- and observation-independent Dec-MDP with joint reward is NP-complete.

## 5 Other subclasses of interactions

As our new results will now show, there is a limit to this sort of complexity reduction: other relatively obvious combinations of independence relationships do not bear the same fruit. That is, we show the NP-completeness result to be specific to fully transition- and observation-independent problems. When these properties are not fully present, worst-case complexity is once again NEXP.

### 5.1 Reward-independent-only models are NEXP-complete

We begin with a result that is rather simple, but has not, to the best of our knowledge, been established before. We consider the *inverse* of the NP-complete problem of Theorem 3: a Dec-MDP with reward-independence (Df. 7), but without transition- or observation-independence (Dfs. 5, 6).

**Theorem 4.** Factored, reward-independent Dec-MDPs with $n$ agents are NEXP-complete.

*Proof Sketch.* For the upper bound, we simply cite Theorem 1, immediately establishing that such problems are in NEXP. For the lower bound, we simply modify the TILING Dec-MDP from Bernstein's reduction proof so as to ensure that the reward-function factors appropriately into strictly local rewards. (*Full details are found in [9], and the supplementary materials, §4.1.*) □

Thus we see that in some respects, transition and observation independence are fundamental to the reduction of worst-case complexity from NEXP to NP. When only the rewards depend upon the actions of both agents, the problems become easier; however, when the situation is reversed,

the general problem remains NEXP-hard. This is not entirely surprising: much of the complexity of planning in decentralized domains stems from the necessity to take account of how one's action-outcomes are affected by the actions of others, and from the complications that ensue when observed information about the system is tied to those actions as well. The structure of rewards, while obviously key to the nature of the optimal (or otherwise) solution, is not as vital—even if agents can separate their individual reward-functions, making them entirely independent, other dependencies can still make the problem extremely complex.

We therefore turn to two other interesting special-case Dec-MDP frameworks, in which independent reward functions are accompanied by restricted degrees of transition- and observation-based interaction. While some empirical evidence has suggested that these problems may be easier on average to solve, nothing has previously been shown about their worst-case complexity. We fill in these gaps, showing that even under such restricted dynamics, the problems remain NEXP-hard.

## 5.2 Event-driven-interaction models are NEXP-complete

The first model we consider is one of Becker et al. [10], which generalizes the notion of a fully transition-independent Dec-MDP. In this model, a set of *primitive events*, consisting of state-action transitions, is defined for each agent. Such events can be thought of as occasions upon which that agent takes the given action to generate the associated state transition. *Dependencies* are then introduced in the form of relationships between one agent's possible actions in given states and another agent's primitive events.

While no precise worst-case complexity results have been previously proven, the authors do point out that the class of problems has an upper-bound deterministic complexity that is exponential in the size of the state space, $|S|$, and doubly exponential in the number of defined interactions. This potentially bad news is mitigated by noting that if the number of interactions is small, then reasonably-sized problems can still be solved. Here, we examine this issue in detail, showing that, in fact these problems are NEXP-hard (indeed, NEXP-complete); however, when the number of dependencies is a log-factor of the size of the problem state-space, worst-case NP-hardness is achieved.

We begin with the formal framework of the model. Again, we give all definitions in terms of Dec-POMDPs; they apply immediately to Dec-MDPs in particular.

**Definition 9** (History). A *history* for an agent $\alpha_i$ in a factored, $n$-agent Dec-POMDP $\mathcal{D}$ is a sequence of possible local states and actions, beginning in the agent's initial state: $\Phi_i = [\hat{s}_i^0, a_i^0, \hat{s}_i^1, a_i^1, \ldots]$. When a problem has a finite time-horizon $T$, all possible complete histories will be of the form $\Phi_i^T = [\hat{s}_i^0, a_i^0, \hat{s}_i^1, a_i^1, \ldots, \hat{s}_i^{T-1}, a_i^T, \hat{s}_i^T]$.

**Definition 10** (Events in a History). A *primitive event* $e = (\hat{s}_i, a_i, \hat{s}_i')$ for an agent $\alpha_i$ is a triple representing a transition between two local states, given some action $a_i \in A_i$. An *event* $E = \{e_1, e_2, \ldots, e_h\}$ is a set of primitive events. A primitive event $e$ *occurs in the history* $\Phi_i$, written $\Phi_i \vDash e$, if and only if the triple $e$ is a sub-sequence of the sequence $\Phi_i$. An event $E$ *occurs in the history* $\Phi_i$, written $\Phi_i \vDash E$, if and only if some component occurs in that history: $\exists e \in E : \Phi_i \vDash e$.

Events can therefore be thought of disjunctively. That is, they specify a set of possible state-action transitions from a Dec-POMDP, local to one of its agents. If the historical sequence of state-action transitions that the agent encounters contains any one of those particular transitions, then the history satisfies the overall event. Events can thus be used, for example, to represent such things as taking a particular action in any one of a number of states over time, or taking one of several actions at some particular state. For technical reasons, namely the use of a specialized solution algorithm, these events are usually restricted in structure, as follows.

**Definition 11** (Proper Events). A primitive event $e$ is *proper* if it occurs at most once in any given history. That is, for any history $\Phi_i$ if $\Phi_i = \Phi_i^1 e \Phi_i^2$ then neither sub-history contains $e$: $\neg(\Phi_i^1 \vDash e) \wedge \neg(\Phi_i^2 \vDash e)$. An event $E$ is *proper* if it consists of proper primitive events that are mutually exclusive, in that no two of them both occur in any history:

$$\forall \Phi_i \neg \exists x, y : (x \neq y) \wedge (e_x \in E) \wedge (e_y \in E) \wedge (\Phi_i \vDash e_x) \wedge (\Phi_i \vDash e_y).$$

Proper primitive events can be used, for instance, to represent actions that take place at particular times (building the time into the local state $\hat{s}_i \in e$). Since any given point in time can only occur once in any history, the events involving such time-steps will be proper by default. A proper event

$E$ can then be formed by collecting all the primitive events involving some single time-step, or by taking all possible primitive events involving an unrepeatable action.

Our new model is then a Dec-MDP with:

1. Two (2) agents.[1]
2. A factored state-space: $S = S_0 \times S_1 \times S_n$.
3. Local full observability: each agent $\alpha_i$ can determine its own portion of the state-space, $\hat{s}_i \in S_0 \times S_i$, exactly.
4. Independent (additive) rewards: $R(\langle s_0, s_1, s_2 \rangle, a_1, a_2) = R_1(\hat{s}_1, a_1) + R_2(\hat{s}_2, a_2)$.

Interactions between agents are given in terms of a set of *dependencies* between certain state-action transitions for one agent, and events featuring transitions involving the other agent. Thus, if a history contains one of the primitive events from the latter set, this can have some direct effect upon the transition-model for the first agent, introducing probabilistic transition-dependencies.

**Definition 12** (Dependency). A *dependency* is a pair $d_{ij}^k = \langle E_i^k, D_j^k \rangle$, where $E_i^k$ is a proper event defined over primitive events for agent $\alpha_i$, and $D_j^k$ is a set of state-action pairs $\langle \hat{s}_j, a_j \rangle$ for agent $\alpha_j$, such that each pair occurs in at most one dependency:

$$\neg (\exists k, k', s_j, a_j) \, (k \neq k') \; \& \; \langle s_j, a_j \rangle \in D_j^k \in d_{ij}^k \; \& \; \langle s_j, a_j \rangle \in D_j^{k'} \in d_{ij}^{k'}.$$

Such a dependency is thus a collection of possible actions that agent $\alpha_j$ can take in one of its local state, each of which depends upon whether the other agent $\alpha_i$ has made one of the state-transitions in its own set of primitive events. Such structures can be used to model, for instance, cases where one agent cannot successfully complete some task until the other agent has completed an enabling sub-task, or where the precise outcome depends upon the groundwork laid by the other agent.

**Definition 13** (Satisfying Dependencies). A dependency $d_{ij}^k = \langle E_i^k, D_j^k \rangle$ is satisfied when the current history for enabling agent $\alpha_i$ contains the relevant event: $\Phi_i \vDash E_i^k$. For any state-action pair $\langle \hat{s}_j, a_j \rangle$, we define a Boolean indicator variable $b_{\hat{s}_j a_j}$, which is true if and only if some dependency that contains the pair is satisfied:

$$b_{\hat{s}_j a_j} = \begin{cases} 1 & \text{if } (\exists d_{ij}^k = \langle E_i^k, D_j^k \rangle) \, \langle \hat{s}_j, a_j \rangle \in D_j^k \; \& \; \Phi_i \vDash E_i^k, \\ 0 & \text{otherwise.} \end{cases}$$

The existence of dependencies allows us to factor the overall state-transition function into two parts, each of which depends only on an agent's local state, action, and relevant indicator variable.

**Definition 14** (Local Transition Function). The transition function for our Dec-MDP is factored into two functions, $P_1$ and $P_2$, each defining the distribution over next possible local states: $P_i(\hat{s}_i' \mid \hat{s}_i, a_i, b_{\hat{s}_i a_i})$. We can thus write $P_i(\hat{s}_i, a_i, b_{\hat{s}_i a_i}, \hat{s}_i')$ for this transition probability.

When agents take some action in a state for which dependencies exist, they observe whether or not the related events have occurred; that is, after taking any action $a_j$ in state $s_j$, they can observe the state of indicator variable $b_{\hat{s}_j a_j}$.

With these definitions in place, we can now show that the worst-case complexity of the event-based problems is the same as the general Dec-POMDP class.

**Theorem 5.** Factored, finite-horizon, $n$-agent Dec-MDPs with local full observability, independent rewards, and event-driven interactions are NEXP-complete.

*Proof Sketch.* Again, the upper bound is immediate from Theorem 1, since the event-based structure is just a specific case of general reward-dependence, and such models can always be converted into Dec-MDPs without any events. For the lower bound, we again provide a reduction from TILING, constrained to our special case. Local reward independence, which was not present in the original problem, is ensured by using event dependencies to affect *future* rewards of the other agent. Thus, local immediate rewards remain dependent only upon the actions of the individual agent, but the state in which that agent finds itself (and so the options available to its reward function) can depend upon events involving the other agent. (*See [9] and supplemental materials, §4.2.*) □

### 5.2.1 A special, NP-hard case

The prior result requires allowing the number of dependencies in the problem to grow as a factor of $\log n$, for a TILING grid of size $(n \times n)$. Since the size of the state-space $S$ in the reduced Dec-MDP is also $\mathcal{O}(\log n)$, the number of dependencies is $\mathcal{O}(|S|)$. Thus, the NEXP-completeness result holds for any event-based Dec-MDP where the number of dependencies is linear in the state-space. When we are able to restrict the number of dependencies further, however, we can do better.

**Theorem 6.** A factored, finite-horizon, $n$-agent Dec-MDP with local full observability, independent rewards, and event-driven interactions are solvable in nondeterministic polynomial time (NP) if the number of dependencies is $\mathcal{O}(\log |S|)$, where $S$ is the state-set of the problem.

*Proof Sketch.* As shown by Becker [10], we can use the Coverage Set algorithm to generate an optimal policy for a problem of this type, in time that is exponential in the number of dependencies. Clearly, if this number is logarithmic in the size of the state-set, then solution time is polynomial in the problem size. (*See [9] and supplemental materials, §4.2.1.*) □

### 5.2.2 Discussion of the results

These results are interesting for two reasons. First, NEXP-completeness of the event-based case, even with independent rewards and local full observability (Theorem 5), means that many interesting problems are potentially intractable. Becker et al. [10] show how to use event-dependencies to represent common structures in the TAEMS task modeling language, used in many real-world domains [11, 12, 13]; our complexity analysis thus extends to such practical problems. Second, isolating where complexity is lower can help determine what task structures and agent interrelationships lead to intractability. In domains where the dependency structure can be kept relatively simple, it may be possible to derive optimal solutions feasibly. Both subjects are worth further study.

### 5.3 State-dependent-action models are NEXP-complete

Guo and Lesser [14, 15, 16] consider another specialized Dec-MDP subclass, with apparently even more restricted types of interaction. Agent state-spaces are again separate, and all action-transitions and rewards are independent. Such problems are not wholly decoupled, however, as the actions available to each agent at any point depend upon the global system state. Thus, agents interact by making choices that restrict or broaden the range of actions available to others.

**Definition 15** (Dec-MDP with State-Dependent Actions). An $n$-agent *Dec-MDP with state-dependent actions* is a tuple $\mathcal{D} = \langle S_0, \{S_i\}, \{A_i\}, \{B_i\}, \{P_i\}, \{R_i\}, T \rangle$, where:

- $S_0$ is a set of shared states, and $S_i$ is the state-space of agent $s_i$, with global state space $S = S_0 \times S_1 \times \cdots \times S_n$, and initial state $s^0 \in S$; each $A_i$ is the action-set for $\alpha_i$; $T \in \mathbb{N}$ is the finite time-horizon of the problem.

- Each $B_i : S \to 2^{A_i}$ is a mapping from *global states* of the system to some set of available actions for each agent $\alpha_i$. For all $s \in S$, $B_i(s) \neq \emptyset$.

- $P_i : (S_0 \times S_i) \times A_i(S_0 \times S_i)$ is the state-transition function over local states for $\alpha_i$. The global transition function is simply the product of individual $P_i$.

- $R_i : (S_0 \times S_i) \to \Re$ is a local reward function for agent $\alpha_i$. We let the global reward function be the sum of local rewards.

Note that there need be no observations in such a problem; given local full observability, each agent observes only its local states. Furthermore, it is presumed that each agent can observe its own available actions in any state; a local policy is thus a mapping from local states to available actions.

For such cases, Guo presents a planning algorithm based on heuristic action-set pruning, along with a learning algorithm. While empirical results show that these methods are capable of solving potentially large instances, we again know very little about the analytical worst-case difficulty of problems with state-dependent actions. An NP-hardness lower bound is given [14] for the overall class, by reducing a normal-form game to the state-dependent model, but this is potentially quite weak, since no upper bound has been established, and even the operative algorithmic complexity of the given solution method is not well understood. We address this situation, showing that the problem is also just as hard as the general case.

**Theorem 7.** Factored, finite-horizon, $n$-agent Dec-MDPs with local full observability, independent rewards, and state-dependent action-sets are NEXP-complete.

*Proof Sketch.* Once more, we rely upon the general upper bound on the complexity of Dec-POMDPs (Theorem 1). The lower bound is by another TILING reduction. Again, we "record" actions of each agent in the state-space of the other, ensuring purely local rewards and local full observability. This time, however, we use the fact that action-sets depend upon the global state (rather than events) to enforce the desired dynamics. That is, we add special state-dependent actions that, based on their availability (or lack thereof), affect each agent's local reward. (*See [9], and supplemental §4.3.*)  □

### 5.3.1  Discussion of the result

Guo and Lesser [16, 14] were able to show that deciding whether a decentralized problem with state-based actions had an equilibrium solution with value greater than $k$ was NP-hard. It was not ascertained whether or not this lower bound was tight, however; this remained a significant open question. Our results show that this bound was indeed too low. Since an optimal joint policy will be an equilibrium for the special case of additive rewards, the general problem can be no easier.

This is interesting, for reasons beyond the formal. Such decentralized problems indeed appear to be quite simple in structure, requiring wholly independent rewards and action-transitions, so that agents can only interact with one another via choices that affect which actions are available. (A typical example involves two persons acting completely regardless of one another, except for the existence of a single rowboat, used for crossing a stream; if either agent uses the rowboat to get to the other side, then that action is no longer available to the other.) Such problems are intuitive, and common, and not all of them are hard to solve, obviously. At the same time, however, our results show that the same structures can be intractable in the worst case, establishing that even seemingly simple interactions between agents can lead to prohibitively high complexity in decentralized problems.

## 6  Conclusions

This work addresses a number of existing models for decentralized problem-solving. In each case, the models restrict agent interaction in some way, in order to produce a special sub-case of the general Dec-POMDP problem. It has been known for some time that systems where agents act entirely independently, but share rewards, have reduced worst-case complexity. We have shown that this does not apply to other variants, where we relax the independence requirements even only a little. In all of the cases addressed, the new problem variants are as hard as the general case. This fact, combined with results showing many other decentralized problem models to be equivalent to the general Dec-POMDP model, or strictly harder [17], reveals the essential difficulty of optimal planning in decentralized settings. Together, these results begin to suggest that optimal solutions to many common multiagent problems must remain out of reach; in turn, this indicates that we must look to approximate or heuristic methods, since such problems are so prevalent in practice.

At the same time, it must be stressed that the NEXP-complexity demonstrated here is a *worst-case* measure. Not all decentralized domains are going to be intractable, and indeed the event-based and action-set models have been shown to yield to specialized solution methods in many cases, enabling us to solve interesting instances in reasonable amounts of time. When the number of action-dependencies is small, or there are few ways that agents can affect available action-sets, it may well be possible to provide optimal solutions effectively. That is, the high worst-case complexity is no guarantee that *average-case* difficulty is likewise high. This remains a vital open problem in the field. While establishing the average case is often difficult, if not impossible—given that the notion of an "average" planning or decision problem is often ill-defined—it is still worth serious consideration.

### Acknowledgments

This material is based upon work supported by the the Air Force Office of Scientific Research under Award No. FA9550-05-1-0254. Any opinions, findings, and conclusions or recommendations expressed in this publication are those of the authors and do not necessarily reflect the views of AFOSR. The first author also acknowledges the support of the Andrew W. Mellon Foundation CTW Computer Science Consortium Fellowship.

## Footnotes

[1]The model can be extended to $n$ agents with little real difficulty. Since we will show that the 2-agent case is NEXP-hard, however, this will suffice for the general claim.

# References

[1] Daniel S. Bernstein, Shlomo Zilberstein, and Neil Immerman. The complexity of decentralized control of Markov decision processes. In *Proceedings of the Sixteenth Conference on Uncertainty in Artificial Intelligence*, pages 32–37, Stanford, California, 2000.

[2] Daniel S. Bernstein, Robert Givan, Neil Immerman, and Shlomo Zilberstein. The complexity of decentralized control of Markov decision processes. *Mathematics of Operations Research*, 27(4):819–840, 2002.

[3] Judy Goldsmith and Martin Mundhenk. Competition adds complexity. In J.C. Platt, D. Koller, Y. Singer, and S. Roweis, editors, *Advances in Neural Information Processing Systems 20*, pages 561–568. MIT Press, Cambridge, MA, 2008.

[4] Harry R. Lewis. Complexity of solvable cases of the decision problem for predicate calculus. In *Proceedings of the Nineteenth Symposium on the Foundations of Computer Science*, pages 35–47, Ann Arbor, Michigan, 1978.

[5] Christos H. Papadimitriou. *Computational Complexity*. Addison-Wesley, Reading, Massachusetts, 1994.

[6] Raphen Becker, Shlomo Zilberstein, Victor Lesser, and Claudia V. Goldman. Transition-independent decentralized Markov decision processes. In *Proceedings of the Second International Joint Conference on Autonomous Agents and Multi-Agent Systems*, pages 41–48, Melbourne, Australia, 2003.

[7] Raphen Becker, Shlomo Zilberstein, Victor Lesser, and Claudia V. Goldman. Solving transition independent decentralized MDPs. *Journal of Artificial Intelligence Research*, 22:423–455, November 2004.

[8] Claudia V. Goldman and Shlomo Zilberstein. Decentralized control of cooperative systems: Categorization and complexity analysis. *Journal of Artificial Intelligence Research*, 22:143–174, 2004.

[9] Martin Allen. *Agent Interactions in Decentralized Environments*. PhD thesis, University of Massachusetts, Amherst, Massachusetts, 2009. Available at `http://scholarworks.umass.edu/open_access_dissertations/1/`.

[10] Raphen Becker, Victor Lesser, and Shlomo Zilberstein. Decentralized Markov decision processes with event-driven interactions. In *Proceedings of the Third International Joint Conference on Autonomous Agents and Multi-Agent Systems*, pages 302–309, New York, New York, 2004.

[11] Keith S. Decker and Victor R. Lesser. Quantitative modeling of complex environments. *International Journal of Intelligent Systems in Accounting, Finance and Management*, 2:215–234, 1993.

[12] V. Lesser, K. Decker, T.Wagner, N. Carver, A. Garvey, B. Horling, D. Neiman, R. Podorozhny, M. Nagendra Prasad, A. Raja, R. Vincent, P. Xuan, and X.Q Zhang. Evolution of the GPGP/TAEMS domain-independent coordination framework. *Autonomous Agents and Multi-Agent Systems*, 9(1):87–143, 2004.

[13] Tom Wagner, Valerie Guralnik, and John Phelps. TAEMS agents: Enabling dynamic distributed supply chain management. *Journal of Electronic Commerce Research and Applications*, 2:114–132, 2003.

[14] AnYuan Guo. *Planning and Learning for Weakly-Coupled Distributed Agents*. PhD thesis, University of Massachusetts, Amherst, 2006.

[15] AnYuan Guo and Victor Lesser. Planning for weakly-coupled partially observable stochastic games. In *Proceedings of the 19th International Joint Conference on Artificial Intelligence*, pages 1715–1716, Edinburgh, Scotland, 2005.

[16] AnYuan Guo and Lesser Victor. Stochastic planning for weakly-coupled distributed agents. In *Proceedings of the Fifth Joint Conference on Autonomous Agents and Multiagent Systems*, pages 326–328, Hakodate, Japan, 2006.

[17] Sven Seuken and Shlomo Zilberstein. Formal models and algorithms for decentralized decision making under uncertainty. *Autonomous Agents and Multi-Agent Systems*, 17(2):190–250, 2008.

